# Kernels for Multi–task Learning

**Charles A. Micchelli**
Department of Mathematics and Statistics
State University of New York,
The University at Albany
1400 Washington Avenue, Albany, NY, 12222, USA

**Massimiliano Pontil**
Department of Computer Sciences
University College London
Gower Street, London WC1E 6BT, England, UK

## Abstract

This paper provides a foundation for multi–task learning using reproducing kernel Hilbert spaces of vector–valued functions. In this setting, the kernel is a matrix–valued function. Some explicit examples will be described which go beyond our earlier results in [7]. In particular, we characterize classes of matrix–valued kernels which are linear and are of the dot product or the translation invariant type. We discuss how these kernels can be used to model relations between the tasks and present linear multi–task learning algorithms. Finally, we present a novel proof of the representer theorem for a minimizer of a regularization functional which is based on the notion of minimal norm interpolation.

## 1 Introduction

This paper addresses the problem of learning a vector–valued function $f : \mathcal{X} \to \mathcal{Y}$, where $\mathcal{X}$ is a set and $\mathcal{Y}$ a Hilbert space. We focus on linear spaces of such functions that admit a reproducing kernel, see [7]. This study is valuable from a variety of perspectives. Our main motivation is the practical problem of multi–task learning where we wish to learn many related regression or classification functions simultaneously, see eg [3, 5, 6]. For instance, image understanding requires the estimation of multiple binary classifiers simultaneously, where each classifier is used to detect a specific object. Specific examples include locating a car from a pool of possibly similar objects, which may include cars, buses, motorbikes, faces, people, etc. Some of these objects or tasks may share common features so it would be useful to relate their classifier parameters. Other examples include multi–modal human computer interface which requires the modeling of both, say, speech and vision, or tumor prediction in bioinformatics from multiple micro–array datasets.

Moreover, the spaces of vector–valued functions described in this paper may be useful for learning continuous transformations. In this case, $\mathcal{X}$ is a space of parameters and $\mathcal{Y}$ a Hilbert space of functions. For example, in face animation $\mathcal{X}$ represents pose and expression of a face and $\mathcal{Y}$ a space of functions $\mathbb{R}^2 \to \mathbb{R}$, although in practice one considers discrete images in which case $f(x)$ is a finite dimensional vector whose components are

associated to the image pixels. Other problems such as image morphing, can be formulated as vector–valued learning.

When $\mathcal{Y}$ is an $n-$dimensional Euclidean space, one straightforward approach in learning a vector–valued function $f = (f_1, \ldots, f_n)$ consists in separately representing each component of $f$ by a linear space of smooth functions and then learn these components independently, for example by minimizing some regularized error functional. This approach does not capture relations between components of $f$ (which are associated to tasks or pixels in the examples above) and should not be the method of choice when these relations occur. In this paper we investigate how kernels can be used for representing vector–valued functions. We proposed to do this by using a matrix–valued kernel $K : \mathcal{X} \times \mathcal{X} \to \mathbb{R}^{n \times n}$ that reflects the interaction amongst the components of $f$. This paper provides a foundation for this approach. For example, in the case of support vector machines (SVM's) [10], appropriate choices of the matrix–valued kernel implement a trade–off between large margin of each per–task SVM and large margin of combinations of these SVM's, eg their average.

The paper is organized as follows. In section 2 we formalize the above observations and show that reproducing Hilbert spaces (RKHS) of vector–valued functions admit a kernel with values which are bounded linear operators on the output space $\mathcal{Y}$ and characterize the form some of these operators in section 3. Finally, in section 4 we provide a novel proof for the representer theorem which is based on the notion of minimal norm interpolation and present linear multi–task learning algorithms.

## 2   RKHS of vector–valued functions

Let $\mathcal{Y}$ be a real Hilbert space with inner product $(\cdot, \cdot)$, $\mathcal{X}$ a set, and $\mathcal{H}$ a linear space of functions on $\mathcal{X}$ with values in $\mathcal{Y}$. We assume that $\mathcal{H}$ is also a Hilbert space with inner product $\langle \cdot, \cdot \rangle$. We present two methods to enhance standard RKHS to vector–valued functions.

### 2.1   Matrix–valued kernels based on Aronszajn

The first approach extends the scalar case, $\mathcal{Y} = \mathbb{R}$, in [2].

**Definition 1** *We say that $\mathcal{H}$ is a reproducing kernel Hilbert space (RKHS) of functions $f : \mathcal{X} \to \mathcal{Y}$, when for any $y \in \mathcal{Y}$ and $x \in \mathcal{X}$ the linear functional which maps $f \in \mathcal{H}$ to $(y, f(x))$ is continuous on $\mathcal{H}$.*

We conclude from the Riesz Lemma (see, e.g., [1]) that, for every $x \in \mathcal{X}$ and $y \in \mathcal{Y}$, there is a linear operator $K_x : \mathcal{Y} \to \mathcal{H}$ such that

$$(y, f(x)) = \langle K_x y, f \rangle. \tag{2.1}$$

For every $x, t \in \mathcal{X}$ we also introduce the linear operator $K(x, t) : \mathcal{Y} \to \mathcal{Y}$ defined, for every $y \in \mathcal{Y}$, by

$$K(x, t)y := (K_t y)(x). \tag{2.2}$$

In the proposition below we state the main properties of the function $K$. To this end, we let $\mathcal{L}(\mathcal{Y})$ be the set of all bounded linear operators from $\mathcal{Y}$ into itself and, for every $A \in \mathcal{L}(\mathcal{Y})$, we denote by $A^*$ its adjoint. We also use $\mathcal{L}_+(\mathcal{Y})$ to denote the cone of positive semidefinite bounded linear operators, i.e. $A \in \mathcal{L}_+(\mathcal{Y})$ provided that, for every $y \in \mathcal{Y}$, $(y, Ay) \geq 0$. When this inequality is strict for all $y \neq 0$ we say $A$ is positive definite. We also denote by $\mathbb{N}_m$ the set of positive integers up to and including $m$. Finally, we say that $\mathcal{H}$ is *normal* provided there does not exist $(x, y) \in \mathcal{X} \times (\mathcal{Y}\backslash\{0\})$ such that the linear functional $(y, f(x)) = 0$ for all $f \in \mathcal{H}$.

**Proposition 1** *If $K(x, t)$ is defined, for every $x, t \in \mathcal{X}$, by equation (2.2) and $K_x$ is given by equation (2.1) then the kernel $K$ satisfies, for every $x, t \in \mathcal{X}$, the following properties:*

(a) *For every $y, z \in \mathcal{Y}$, we have that $(y, K(x,t)z) = \langle K_t z, K_x y \rangle$.*

(b) $K(x,t) \in \mathcal{L}(\mathcal{Y})$, $K(x,t) = K(t,x)^*$, *and* $K(x,x) \in \mathcal{L}_+(\mathcal{Y})$.

   *Moreover, $K(x,x)$ is positive definite for all $x \in \mathcal{X}$ if and only if $\mathcal{H}$ is normal.*

(c) *For any $m \in \mathbb{N}$, $\{x_j : j \in \mathbb{N}_m\} \subseteq \mathcal{X}$, $\{y_j : j \in \mathbb{N}_m\} \subseteq \mathcal{Y}$ we have that*

$$\sum_{j, \ell \in \mathbb{N}_m} (y_j, K(x_j, x_\ell)y_\ell) \geq 0. \tag{2.3}$$

PROOF.  We prove (a) by merely choosing $f = K_t z$ in equation (2.1) to obtain that

$$\langle K_x y, K_t z \rangle = (y, (K_t z)(x)) = (y, K(x,t)z). \tag{2.4}$$

Consequently, from this equation, we conclude that $K(x,t)$ admits an algebraic adjoint $K(t,x)$ defined everywhere on $\mathcal{Y}$ and, so, the uniform boundness principle, see, eg, [1, p. 48] implies that $K(x,t) \in \mathcal{L}(\mathcal{Y})$ and $K(x,t) = K(t,x)^*$. Moreover, choosing $t = x$ in (a) proves that $K(x,x) \in \mathcal{L}_+(\mathcal{Y})$. As for the positive definiteness of $K(x,x)$, merely use equations (2.1) and property (a). These remarks prove (b). As for (c), we again use property (a) to obtain that

$$\sum_{j, \ell \in \mathbb{N}_m} (y_j, K(x_j, x_\ell)y_\ell) = \sum_{j, \ell \in \mathbb{N}_m} \langle K_{x_j} y_j, K_{x_\ell} y_\ell \rangle = \| \sum_{j \in \mathbb{N}_m} K_{x_j} y_j \|^2 \geq 0.$$

This completes the proof.

For simplicity, we say that $K : \mathcal{X} \times \mathcal{X} \to \mathcal{L}(\mathcal{Y})$ is a *matrix–valued kernel* (or simply a kernel if no confusion will arise) if it satisfies properties (b) and (c). So far we have seen that if $\mathcal{H}$ is a RKHS of vector–valued functions, there exists a kernel. In the spirit of the Moore-Aronszajn's theorem for RKHS of scalar functions [2], it can be shown that if $K : \mathcal{X} \times \mathcal{X} \to \mathcal{L}(\mathcal{Y})$ is a kernel then there exists a unique (up to an isometry) RKHS of functions from $\mathcal{X}$ to $\mathcal{Y}$ which admits $K$ as the reproducing kernel. The proof parallels the scalar case.

Given a vector–valued function $f : \mathcal{X} \to \mathcal{Y}$ we associate to it a scalar–valued function $F : \mathcal{X} \times \mathcal{Y} \to \mathbb{R}$ defined by

$$F(x, \lambda) := (\lambda, f(x)), \quad x \in \mathcal{X}, \lambda \in \mathcal{Y}. \tag{2.5}$$

We let $\mathcal{H}^1$ be the linear space of all such functions. Thus, $\mathcal{H}^1$ consists of functions which are *linear* in their second variable. We make $\mathcal{H}^1$ into a Hilbert space by choosing $\|F\| = \|f\|$. It then follows that $\mathcal{H}^1$ is a RKHS with reproducing *scalar–valued* kernel defined, for all $(x,y), (t,z) \in \mathcal{X} \times \mathcal{Y}$, by the formula

$$K^1((x,y),(t,z)) := (y, K(x,t)z). \tag{2.6}$$

## 2.2   Feature map

The second approach uses the notion of feature map, see e.g. [9]. A feature map is a function $\Phi : \mathcal{X} \times \mathcal{Y} \to \mathcal{W}$ where $\mathcal{W}$ is a Hilbert space. A feature map representation of a kernel $K$ has the property that, for every $x, t \in \mathcal{X}$ and $y, z \in \mathcal{Y}$ there holds the equation

$$(\Phi(x,y), \Phi(t,z)) = (y, K(x,t)z).$$

From equation (2.4) we conclude that every kernel admits a feature map representation (a Mercer type theorem) with $\mathcal{W} = \mathcal{H}$. With additional hypotheses on $\mathcal{H}$ and $\mathcal{Y}$ this representation can take a familiar form

$$K_{\ell q}(x,t) = \sum_{r \in \mathbb{N}} \Phi_r^\ell(x) \Phi_r^q(t), \quad \ell, q \in \mathbb{N}. \tag{2.7}$$

Much more importantly, we begin with a feature map $\Phi(x, \lambda) = ((\Phi^\ell(x), \lambda) : \ell \in \mathbb{N})$ where $\lambda \in \mathcal{W}$, this being the space of squared summable sequence on $\mathbb{N}$. We wish to learn a function $f : \mathcal{X} \to \mathcal{Y}$ where $\mathcal{Y} = \mathcal{W}$ and $f = (f_\ell : \ell \in \mathbb{N})$ with $f_\ell = (w, \Phi^\ell) := \sum_{r \in \mathbb{N}} w_r \Phi_r^\ell$ for each $\ell \in \mathbb{N}$, where $w \in \mathcal{W}$. We choose $\|f\| = \|w\|$ and conclude that the space of all such functions is a Hilbert space of function from $\mathcal{X}$ to $\mathcal{Y}$ with kernel (2.7). These remarks connect feature maps to kernels and vice versa. Note a kernel may have many maps which represent it and a feature map representation for a kernel may not be the appropriate way to write it for numerical computations.

# 3 Kernel construction

In this section we characterize a wide variety of kernels which are potentially useful for applications.

## 3.1 Linear kernels

A first natural question concerning RKHS of vector–valued functions is: if $\mathcal{X}$ is $\mathbb{R}^d$ what is the form of linear kernels? In the scalar case a linear kernel is a quadratic form, namely $K(x, t) = (x, Qt)$, where $Q$ is a $d \times d$ positive semidefinite matrix. We claim that for $\mathcal{Y} = \mathbb{R}^n$ any linear matrix–valued kernel $K = (K_{\ell q} : \ell, q \in \mathbb{N}_n)$ has the form

$$K_{\ell q}(x, t) = (B_\ell x, B_q t), \quad x, t \in \mathbb{R}^d \tag{3.8}$$

where $B_\ell$ are $p \times d$ matrices for some $p \in \mathbb{N}$. To see that such $K$ is a kernel simply note that $K$ is in the Mercer form (2.7) for $\Phi_\ell(x) = B_\ell x$. On the other hand, since any linear kernel has a Mercer representation with linear features, we conclude that all linear kernels have the form (3.8). A special case is provided by choosing $p = d$ and $B_\ell$ to be diagonal matrices.

We note that the theory presented in section 2 can be naturally extended to the case where each component of the vector–valued function has a different input domain. This situation is important in multi–task learning, see eg [5]. To this end, we specify sets $\mathcal{X}_\ell$, $\ell \in \mathbb{N}_n$, functions $g_\ell : \mathcal{X}_\ell \to \mathbb{R}$, and note that multi–task learning can be placed in the above framework by defining the input space

$$\mathcal{X} := \mathcal{X}_1 \times \mathcal{X}_2 \times \cdots \times \mathcal{X}_n.$$

We are interested in vector–valued functions $f : \mathcal{X} \to \mathbb{R}^n$ whose coordinates are given by $f_\ell(x) = g_\ell(P_\ell x)$, where $x = (x_\ell : x_\ell \in \mathcal{X}_\ell, \ell \in \mathbb{N}_n)$ and $P_\ell : \mathcal{X} \to \mathcal{X}_\ell$ is a projection operator defined, for every $x \in \mathcal{X}$ by $P_\ell(x) = x_\ell$, $\ell \in \mathbb{N}_n$. For $\ell, q \in \mathbb{N}_n$, we suppose kernel functions $C_{\ell q} : \mathcal{X}_\ell \times \mathcal{X}_q \to \mathbb{R}$ are given such that the matrix valued kernel whose elements are defined as

$$K_{\ell q}(x, t) := C_{\ell q}(P_\ell x, P_q t), \quad \ell, q \in \mathbb{N}_n$$

satisfies properties (b) and (c) of Proposition 1. An example of this construction is provided again by linear functions. Specifically, we choose $\mathcal{X}_\ell = \mathbb{R}^{d_\ell}$, where $d_\ell \in \mathbb{N}$ and $C_{\ell q}(x_\ell, t_q) = (Q_\ell x_\ell, Q_q t_q)$, $x_\ell \in \mathcal{X}_\ell, t_q \in \mathcal{X}_q$, where $Q_\ell$ are $p \times d_\ell$ matrices. In this case, the matrix–valued kernel $K = (K_{\ell q} : \ell, q \in \mathbb{N}_n)$ is given by

$$K_{\ell q}(x, t) = (Q_\ell P_\ell x, Q_q P_q t) \tag{3.9}$$

which is of the form in equation (3.8) for $B_\ell = Q_\ell P_\ell$, $\ell \in \mathbb{N}_n$.

## 3.2 Combinations of kernels

The results in this section are based on a lemma by Schur which state that the elementwise product of two positive semidefinite matrices is also positive semidefinite, see [2, p. 358].

This result implies that, when $\mathcal{Y}$ is finite dimensional, the elementwise product of two matrix–valued kernels is also a matrix–valued kernel. Indeed, in view of the discussion at the end of section 2.2 we immediately conclude the following two lemma hold.

**Lemma 1** *If $\mathcal{Y} = \mathbb{R}^n$ and $K_1$ and $K_2$ are matrix–valued kernels then their elementwise product is a matrix–valued kernel.*

This result allows us, for example, to enhance the linear kernel (3.8) to a polynomial kernel. In particular, if $r$ is a positive integer, we define, for every $\ell, q \in \mathbb{N}_n$,
$$K_{\ell q}(x,t) := (B_\ell x_\ell, B_q t_q)^r$$
and conclude that $K = (K_{\ell q} : \ell, q \in \mathbb{N}_n)$ is a kernel.

**Lemma 2** *If $G : \mathbb{R}^d \times \mathbb{R}^d \to \mathbb{R}$ is a kernel and $z_\ell : \mathcal{X} \to \mathbb{R}^d$ a vector–valued function, for $\ell \in \mathbb{N}_n$ then the matrix–valued function $K : \mathcal{X} \times \mathcal{X} \to \mathbb{R}^{n \times n}$ whose elements are defined, for every $x, t \in \mathcal{X}$, by*
$$K_{\ell q}(x,t) = G(z_\ell(x), z_q(t))$$
*is a matrix–valued kernel.*

This lemma confirms, as a special case, that if $z_\ell(x) = B_\ell x$ with $B_\ell$ a $p \times d$ matrix, $\ell \in \mathbb{N}_n$, and $G : \mathbb{R}^d \times \mathbb{R}^d \to \mathbb{R}$ is a scalar–valued kernel, then the function (3.8) is a matrix–valued kernel. When $G$ is chosen to be a Gaussian kernel, we conclude that $K_{\ell q}(x,t) = \exp(-\sigma \|B_\ell x - B_q t\|^2)$ is a matrix–valued kernel.

In the scalar case it is well–known that a nonnegative combination of kernels is a kernel. The next proposition extends this result to matrix–valued kernels.

**Proposition 2** *If $K_j, j \in \mathbb{N}_s$, $s \in \mathbb{N}$ are scalar–valued kernels and $A_j \in \mathcal{L}_+(\mathcal{Y})$ then the function*
$$K = \sum_{j \in \mathbb{N}_s} A_j K_j \tag{3.10}$$
*is a matrix–valued kernel.*

PROOF.    For any $x, t \in \mathcal{X}$ and $c, d \in \mathcal{Y}$ we have that
$$(c, K(x,t)z) = \sum_{j \in \mathbb{N}_s} (c, A_j d) K_j(x,t)$$
and so the proposition follows form the Schur lemma.

Other results of this type can be found in [7]. The formula (3.10) can be used to generate a wide variety of matrix–valued kernels which have the flexibility needed for learning. For example, we obtain polynomial matrix–valued kernels by setting $\mathcal{X} = \mathbb{R}^d$ and $K_j(x,t) = (x,t)^j$, where $x, t \in \mathbb{R}^d$. We remark that, generally, the kernel in equation (3.10) *cannot* be reduced to a diagonal kernel. An interesting case of Proposition 2 is provided by *low rank* kernels which may be useful in situations where the components of $f$ are *linearly related*, that is, for every $f \in \mathcal{H}$ and $x \in \mathcal{X}$ $f(x)$ lies in a linear subspace $\mathcal{M} \subseteq \mathcal{Y}$. In this case, it is desirable to use a kernel which has the same property that $f(x) \in \mathcal{M}$, $x \in \mathcal{X}$ for all $f \in \mathcal{H}$. We can ensure this by an appropriate choice of the matrices $A_j$. For example, if $\mathcal{M} = span(\{b_j : j \in \mathbb{N}_s\})$ we may choose $A_j = b_j b_j^*$.

Matrix–valued Gaussian mixtures are obtained by choosing $\mathcal{X} = \mathbb{R}^d$, $\mathcal{Y} = \mathbb{R}^n$, $\{\sigma_j : j \in \mathbb{N}_s\} \subset \mathbb{R}_+$, and $K_j(x,t) = \exp(-\sigma_j \|x - t\|^2)$. Specifically,
$$K(x,t) = \sum_{j \in \mathbb{N}_s} A_j e^{-\sigma_j \|x-t\|^2}$$
is a kernel on $\mathcal{X} \times \mathcal{X}$ for any $\{A_j : j \in \mathbb{N}_s\} \subseteq \mathcal{L}_+(\mathbb{R}^n)$.

# 4 Regularization and minimal norm interpolation

Let $V : \mathcal{Y}^m \times \mathbb{R}_+ \to \mathbb{R}$ be a prescribed function and consider the problem of minimizing the functional

$$E(f) := V\left((f(x_j) : j \in \mathbb{N}_m), \|f\|^2\right) \tag{4.11}$$

over all functions $f \in \mathcal{H}$. A special case is covered by the functional of the form

$$E(f) := \sum_{j \in \mathbb{N}_m} Q(y_j, f(x_j)) + \gamma \|f\|^2 \tag{4.12}$$

where $\gamma$ is a positive parameter and $Q : \mathcal{Y} \times \mathcal{Y} \to \mathbb{R}_+$ is some prescribed *loss function*, eg the square loss. Within this general setting we provide a "representer theorem" for any function which minimizes the functional in equation (4.11). This result is well-known in the scalar case. Our proof technique uses the idea of minimal norm interpolation, a central notion in function estimation and interpolation.

**Lemma 3** *If $y \in \{(f(x_j) : j \in \mathbb{N}_m) : f \in \mathcal{H}\} \subset \mathbb{R}^m$ the minimum of problem*

$$\min\left\{\|f\|^2 : f(x_j) = y_j,\ j \in \mathbb{N}_m\right\} \tag{4.13}$$

*is unique and admits the form $\hat{f} = \sum_{j \in \mathbb{N}_m} K_{x_j} c_j$.*

We refer to [7] for a proof. This approach achieves both simplicity and generality. For example, it can be extended to normed linear spaces, see [8]. Our next result establishes that the form of any *local* minimizer[1] indeed has the same form as in Lemma 3. This result improves upon [9] where it is proven only for a global minimizer.

**Theorem 1** *If for every $y \in \mathcal{Y}^m$ the function $h : \mathbb{R}_+ \to \mathbb{R}_+$ defined for $t \in \mathbb{R}_+$ by $h(t) := V(y, t)$ is strictly increasing and $f_0 \in \mathcal{H}$ is a local minimum of $E$ then $f_0 = \sum_{j \in \mathbb{N}_m} K_{x_j} c_j$ for some $\{c_j : j \in \mathbb{N}_m\} \subseteq \mathcal{Y}$.*

*Proof:* If $g$ is any function in $\mathcal{H}$ such that $g(x_j) = 0, j \in \mathbb{N}_m$ and $t$ a real number such that $|t| \|g\| \leq \epsilon$, for $\epsilon > 0$, then

$$V\left(y_0, \|f_0\|^2\right) \leq V\left(y_0, \|f_0 + tg\|^2\right).$$

Consequently, we have that $\|f_0\|^2 \leq \|f_0 + tg\|^2$ from which it follows that $(f_0, g) = 0$. Thus, $f_0$ satisfies

$$\|f_0\| = \min\{\|f\| : f(x_j) = f_0(x_j), j \in \mathbb{N}_m,\ f \in \mathcal{H}\}$$

and the result follows from Lemma 3.

## 4.1 Linear regularization

We comment on regularization for linear multi–task learning and therefore consider minimizing the functional

$$R_0(w) := \sum_{j \in \mathbb{N}_m} \sum_{\ell \in \mathbb{N}_n} Q(y_{j\ell}, (w, B_\ell x_j)) + \gamma \|w\|^2 \tag{4.14}$$

for $w \in \mathbb{R}^p$. We set $u_\ell = B_\ell^* w$, $u = (u_\ell : \ell \in \mathbb{N}_n)$, and observe that the above functional is related to the functional

$$R_1(u) := \sum_{j \in \mathbb{N}_m} \sum_{\ell \in \mathbb{N}_n} Q(y_{j\ell}, (u_\ell, x_j)) + \gamma J(u) \tag{4.15}$$

where we have defined the minimum norm functional

$$J(u) := \min\{\|w\|^2 : w \in \mathbb{R}^p, B_\ell^* w = u_\ell, \ell \in \mathbb{N}_n\}. \qquad (4.16)$$

Specifically, we have

$$\min\{R_0(w) : w \in \mathbb{R}^p\} = \min\{R_1((B_\ell w : \ell \in \mathbb{N}_n)) : w \in \mathbb{R}^p\}.$$

The optimal solution $\hat{w}$ of problem (4.16) is given by $\hat{w} = \sum_{\ell \in \mathbb{N}_n} B_\ell c_\ell$, where the vectors $\{c_\ell : \ell \in \mathbb{N}_n\} \subseteq \mathbb{R}^d$ satisfy the linear equations

$$\sum_{k \in \mathbb{N}_n} B_\ell^* B_k c_k = u_\ell, \ \ \ell \in \mathbb{N}_n$$

and

$$J(u) = \sum_{\ell,q \in \mathbb{N}_n} (u_\ell, \tilde{B}_{\ell q}^{-1} u_q)$$

provided the $d \times d$ block matrix $\tilde{B} = (B_\ell^* B_q : \ell, q \in \mathbb{N}_n)$ is nonsingular. We note that this analysis can be extended to the case of different inputs across the tasks by replacing $x_j$ in equations (4.14) and (4.15) by $x_{j,\ell} \in \mathbb{R}^{d_\ell}$ and matrix $B_\ell$ by $Q_\ell P_\ell$, see section 3.1 for the definition of these quantities.

As a special example we choose $B_\ell$ to be the $(n + 1)d \times d$ matrix whose $d \times d$ blocks are all zero expect for the $1-$st and $(\ell + 1)-$th block which are equal to $c^{-1} I_d$ and $I_d$ respectively, where $c > 0$ and $I_d$ is the $d-$dimensional identity matrix. From equation (3.8) the matrix–valued kernel $K$ in equation (3.8) reduce to

$$K_{\ell q}(x,t) = (\frac{1}{c^2} + \delta_{\ell q})(x,t), \ \ \ell, q \in \mathbb{N}_n, \ x,t \in \mathbb{R}^n. \qquad (4.17)$$

Moreover, in this case the minimization in (4.16) is given by

$$J(u) = \frac{c^2}{n + c^2} \sum_{\ell \in \mathbb{N}_n} \|u_\ell\|^2 + \frac{n}{n + c^2} \sum_{\ell \in \mathbb{N}_n} \|u_\ell - \frac{1}{n} \sum_{q \in \mathbb{N}_n} u_q\|^2. \qquad (4.18)$$

The model of minimizing (4.14) was proposed in [6] in the context of support vector machines (SVM's) for these special choice of matrices. The derivation presented here improves upon it. The regularizer (4.18) forces a trade–off between a desirable small size for per–task parameters and closeness of each of these parameters to their average. This trade-off is controlled by the coupling parameter $c$. If $c$ is small the tasks parameters are *related* (closed to their average) whereas a large value of $c$ means the task are learned independently. For SVM's, $Q$ is the Hinge loss function defined by $Q(a,b) := \max(0, 1 - ab)$, $a, b \in \mathbb{R}$. In this case the above regularizer trades off large margin of each per–task SVM with closeness of each SVM to the average SVM. Numerical experiments showing the good performance of the multi–task SVM compared to both independent per–task SVM's (ie, $c = \infty$ in equation (4.17)) and previous multi–task learning methods are also discussed in [6].

The analysis above can be used to derive other linear kernels. This can be done by either introducing the matrices $B_\ell$ as in the previous example, or by modifying the functional on the right hand side of equation (4.15). For example, we choose an $n \times n$ symmetric matrix $A$ all of whose entries are in the unit interval, and consider the regularizer

$$J(u) := \frac{1}{2} \sum_{\ell,q \in \mathbb{N}_n} \|u_\ell - u_q\|^2 A_{\ell q} = \sum_{\ell,q \in \mathbb{N}_n} (u_\ell, u_q) L_{\ell q} \qquad (4.19)$$

where $L = D - A$ with $D_{\ell q} = \delta_{\ell q} \sum_{h \in \mathbb{N}_n} A_{\ell h}$. The matrix $A$ could be the weight matrix of a graph with $n$ vertices and $L$ the graph Laplacian, see eg [4]. The equation $A_{\ell q} = 0$

means that tasks $\ell$ and $q$ are not related, whereas $A_{\ell q} = 1$ means strong relation. In order to derive the matrix–valued kernel we note that (4.19) can be written as $(u, \tilde{L}, u)$ where $\tilde{L}$ is the $n \times n$ block matrix whose $\ell, q$ block is the $d \times d$ matrix $I_d L_{\ell q}$. Thus, we define $w = \tilde{L}^{\frac{1}{2}} u$ so that we have $u_\ell = P_\ell \tilde{L}^{-\frac{1}{2}} w$ (here $L^{-1}$ is the pseudoinverse), where $P_\ell$ is a projection matrix from $\mathbb{R}^{dn}$ to $\mathbb{R}^d$. Consequently, the feature map in equation (2.7) is given by $\Phi^\ell = B_\ell = \tilde{L}^{-\frac{1}{2}} P_\ell^*$ and we conclude that

$$K_{\ell q}(x, t) = (x, P_\ell \tilde{L}^{-1} P_q^* t).$$

Finally, as discussed in section 3.2 one can form polynomials or non-linear functions of the above linear kernels. From Theorem 1 the minimizer of (4.12) is still a linear combination of the kernel at the given data examples.

## 5  Conclusions and future directions

We have described reproducing kernel Hilbert spaces of vector–valued functions and discussed their use in multi–task learning. We have provided a wide class of matrix–valued kernels which should proved useful in applications. In the future it would be valuable to study learning methods, using convex optimization or MonteCarlo integration, for choosing the matrix–valued kernel. This problem seems more challenging that its scalar counterpart due to the possibly large dimension of the output space. Another important problem is to study error bounds for learning in these spaces. Such analysis can clarify the role played by the spectra of the matrix–valued kernel. Finally, it would be interesting to link the choice of matrix–valued kernels to the notion of relatedness between tasks discussed in [5].

### Acknowledgments

This work was partially supported by EPSRC Grant GR/T18707/01 and NSF Grant No. ITR-0312113. We are grateful to Zhongying Chen, Head of the Department of Scientific Computation at Zhongshan University for providing both of us with the opportunity to complete this work in a scientifically stimulating and friendly environment. We also wish to thank Andrea Caponnetto, Sayan Mukherjee and Tomaso Poggio for useful discussions.

## Footnotes

[1] A function $f_0 \in \mathcal{H}$ is a local minimum for $E$ provided that there is a positive number $\epsilon$ such that whenever $f \in \mathcal{H}$ satisfies $\|f_0 - f\| \leq \epsilon$ then $E(f_0) \leq E(f)$.

### References

[1]  N.I. Akhiezer and I.M. Glazman. *Theory of linear operators in Hilbert spaces*, volume I. Dover reprint, 1993.

[2]  N. Aronszajn. Theory of reproducing kernels. *Trans. AMS*, 686:337–404, 1950.

[3]  J. Baxter. A Model for Inductive Bias Learning. *Journal of Artificial Intelligence Research, 12, p. 149– 198*, 2000.

[4]  M. Belkin and P. Niyogi. Laplacian Eigenmaps for Dimensionality Reduction and data Representation *Neural Computation*, 15(6):1373–1396, 2003.

[5]  S. Ben-David and R. Schuller. Exploiting Task Relatedness for Multiple Task Learning. Proc. of the 16–th Annual Conference on Learning Theory (COLT'03), 2003.

[6]  T. Evgeniou and M.Pontil. Regularized Multitask Learning. Proc. of 17-th SIGKDD Conf. on Knowledge Discovery and Data Mining, 2004.

[7]  C.A. Micchelli and M. Pontil. On Learning Vector-Valued Functions. *Neural Computation*, 2004 (to appear).

[8]  C.A. Micchelli and M. Pontil. A function representation for learning in Banach spaces. Proc. of the 17–th Annual Conf. on Learning Theory (COLT'04), 2004.

[9]  B. Schölkopf, R. Herbrich, and A.J. Smola. A Generalized Representer Theorem. Proc. of the 14-th Annual Conf. on Computational Learning Theory (COLT'01), 2001.

[10]  V. N. Vapnik. *Statistical Learning Theory*. Wiley, New York, 1998.
